# The $g$ Factor: Relating Distributions on Features to Distributions on Images

**James M. Coughlan and A. L. Yuille**
Smith-Kettlewell Eye Research Institute,
2318 Fillmore Street,
San Francisco, CA 94115, USA.
Tel. (415) 345-2146/2144. Fax. (415) 345-8455.
Email: coughlan@ski.org, yuille@ski.org

## Abstract

We describe the $g$-factor, which relates probability distributions on image features to distributions on the images themselves. The $g$-factor *depends only on our choice of features and lattice quantization* and is independent of the training image data. We illustrate the importance of the $g$-factor by analyzing how the parameters of Markov Random Field (i.e. Gibbs or log-linear) probability models of images are learned from data by maximum likelihood estimation. In particular, we study homogeneous MRF models which learn image distributions in terms of clique potentials corresponding to feature histogram statistics (cf. Minimax Entropy Learning (MEL) by Zhu, Wu and Mumford 1997 [11]). We first use our analysis of the $g$-factor to determine when the clique potentials decouple for different features. Second, we show that clique potentials can be computed analytically by approximating the $g$-factor. Third, we demonstrate a connection between this approximation and the Generalized Iterative Scaling algorithm (GIS), due to Darroch and Ratcliff 1972 [2], for calculating potentials. This connection enables us to use GIS to improve our multinomial approximation, using Bethe-Kikuchi[8] approximations to simplify the GIS procedure. We support our analysis by computer simulations.

## 1 Introduction

There has recently been a lot of interest in learning probability models for vision. The most common approach is to learn histograms of filter responses or, equivalently, to learn *probability distributions on features* (see right panel of figure (1)). See, for example, [6], [5], [4]. (In this paper the features we are considering will be extracted from the image by filters – hence we use the terms "features" and "filters" synonymously.)

An alternative approach, however, is to learn probability distributions *on the images themselves*. The Minimax Entropy Learning (MEL) theory [11] uses the maximum entropy principle to learn MRF distributions in terms of clique potentials determined by the feature statistics (i.e. histograms of filter responses). (We note that the maximum entropy principle is equivalent to performing maximum likelihood estimation on an MRF whose form is determined by the choice of feature statistics.) When applied to texture modeling it gives a way to unify the filter based approaches (which are often very effective) with the MRF distribution approaches (which are theoretically attractive).

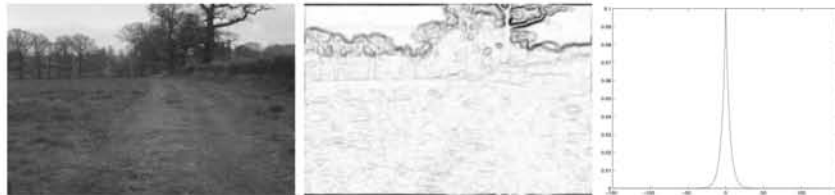

Figure 1: Distributions on images vs. distributions on features. Left and center panels show a natural image and its image gradient magnitude map, respectively. Right panel shows the empirical histogram (i.e. a distribution on a feature) of the image gradient across a dataset of natural images. This feature distribution can be used to create a MRF distribution over images[10]. This paper introduces the $g$-factor to examine connections between the distribution over images and the distribution over features.

As we describe in this paper (see figure (1)), distributions on images and on features can be related by a $g$-factor (such factors arise in statistical physics, see [3]). Understanding the $g$-factor allows us to *approximate it in a form that helps explain why the clique potentials learned by MEL take the form that they do* as functions of the feature statistics. Moreover, *the MEL clique potentials for different features often seem to be decoupled and the $g$-factor can explain why, and when, this occurs.* (I.e. the two clique potentials corresponding to two features $A$ and $B$ are identical whether we learn them jointly or independently).

The $g$-factor is determined only by the form of the features chosen and *the spatial lattice and quantization of the image gray-levels*. It is completely independent of the training image data. It should be stressed that the choice of image lattice, gray-level quantization and histogram quantization can make a big difference to the $g$-factor and hence to the probability distributions which are the output of MEL.

In Section (2), we briefly review Minimax Entropy Learning. Section (3) introduces the $g$-factor and determines conditions for when clique potentials are decoupled. In Section (4) we describe a simple approximation which enables us to learn the clique potentials analytically, and in Section (5) we discuss connections between this approximation and the Generalized Iterative Scaling (GIS) algorithm.

## 2   Minimax Entropy Learning

Suppose we have training image data which we assume has been generated by an (unknown) probability distribution $P_T(\vec{x})$ where $\vec{x}$ represents an image. Minimax Entropy Learning (MEL) [11] approximates $P_T(\vec{x})$ by selecting the distribution with

maximum entropy constrained by observed feature statistics $\vec{\phi}(\vec{x}) = \vec{\psi}_{obs}$. This gives $P(\vec{x}|\vec{\lambda}) = \frac{e^{\vec{\lambda}\cdot\vec{\phi}(\vec{x})}}{Z[\vec{\lambda}]}$, where $\vec{\lambda}$ is a parameter chosen such that $\sum_{\mathbf{x}} P(\vec{x}|\lambda)\phi(\vec{x}) = \vec{\psi}_{obs}$. Or equivalently, so that $\frac{\partial \log Z[\vec{\lambda}]}{\partial \vec{\lambda}} = \vec{\psi}_{obs}$.

We will treat the special case where the statistics $\vec{\phi}$ are the histogram of a shift-invariant filter $\{f_i(\vec{x}) : i = 1, ..., N\}$, where $N$ is the total number of pixels in the image. So $\psi_a = \phi_a(\vec{x}) = \frac{1}{N}\sum_{i=1}^{N} \delta_{a,f_i(\vec{x})}$ where $a = 1, ..., Q$ indicates the (quantized) filter response values. The potentials become $\vec{\lambda}\cdot\vec{\phi}(\vec{x}) = \frac{1}{N}\sum_{a=1}^{Q}\sum_{i=1}^{N} \lambda(a)\delta_{a,f_i(\vec{x})} = \frac{1}{N}\sum_{i=1}^{N} \lambda(f_i(\vec{x}))$. Hence $P(\vec{x}|\vec{\lambda})$ becomes a MRF distribution with clique potentials given by $\lambda(f_i(\vec{x}))$. This determines a Markov random field with the clique structure given by the filters $\{f_i\}$.

MEL also has a feature selection stage based on Minimum Entropy to determine which features to use in the Maximum Entropy Principle. The features are evaluated by computing the entropy $-\sum_{\vec{x}} P(\vec{x}|\vec{\lambda}) \log P(\vec{x}|\vec{\lambda})$ for each choice of features (with small entropies being preferred). A *filter pursuit* procedure was described to determine which filters/features should be considered (our approximations work for this also).

## 3   The $g$-Factor

This section defines the $g$-factor and starts investigating its properties in subsection (3.1). In particular, when, and why, do clique potentials decouple? More precisely, when do the potentials for filters $A$ and $B$ learned simultaneously differ from the potentials for the two filters when they are learned independently?

We address these issues by introducing the $g$-factor $g(\vec{\psi})$ and the associated distribution $\hat{P}_0(\vec{\psi})$:

$$g(\vec{\psi}) = \sum_{\vec{x}} \delta_{\vec{\phi}(\vec{x}),\vec{\psi}}, \quad \hat{P}_0(\vec{\psi}) = \frac{1}{L^N}g(\vec{\psi}). \tag{1}$$

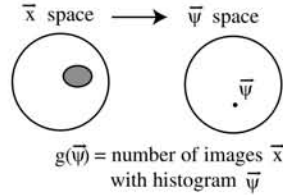

Figure 2: The $g$-factor $g(\vec{\psi})$ counts the number of images $\vec{x}$ that have statistics $\vec{\psi}$. Note that the $g$-factor depends only on the choice of filters and is independent of the training image data.

Here $L$ is the number of grayscale levels of each pixel, so that $L^N$ is the total number of possible images. The $g$-factor is essentially a combinational factor which counts the number of ways that one can obtain statistics $\vec{\psi}$, see figure (2). Equivalently, $\hat{P}_0$ is the default distribution on $\vec{\psi}$ if the images are generated by white noise (i.e. completely random images).

We can use the $g$-factor to compute the induced distribution $\hat{P}(\vec{\psi}|\vec{\lambda})$ on the statistics determined by MEL:

$$\hat{P}(\vec{\psi}|\vec{\lambda}) = \sum_{\vec{x}} \delta_{\vec{\psi},\vec{\phi}(\vec{x})} P(\vec{x}|\vec{\lambda}) = \frac{g(\vec{\psi})e^{\vec{\lambda}\cdot\vec{\psi}}}{Z[\vec{\lambda}]}, \quad Z[\vec{\lambda}] = \sum_{\vec{\psi}} g(\vec{\psi})e^{\vec{\lambda}\cdot\vec{\psi}}. \quad (2)$$

Observe that both $\hat{P}(\vec{\psi}|\vec{\lambda})$ and $\log Z[\vec{\lambda}]$ are sufficient for computing the parameters $\vec{\lambda}$. The $\vec{\lambda}$ can be found by solving either of the following two (equivalent) equations: $\sum_{\vec{\psi}} \hat{P}(\vec{\psi}|\vec{\lambda})\vec{\psi} = \vec{\psi}_{obs}$, or $\frac{\partial \log Z[\vec{\lambda}]}{\partial \vec{\lambda}} = \vec{\psi}_{obs}$, which shows that *knowledge of the g-factor and $e^{\vec{\lambda}\cdot\vec{\psi}}$ are all that is required to do MEL.*

Observe from equation (2) that we have $\hat{P}(\vec{\psi}|\vec{\lambda} = 0) = P_0(\vec{\psi})$. In other words, setting $\vec{\lambda} = 0$ corresponds to a uniform distribution on the images $\vec{x}$.

## 3.1 Decoupling Filters

We now derive an important property of the minimax entropy approach. As mentioned earlier, it often seems that the potentials for filters $A$ and $B$ decouple. In other words, if one applies MEL to two filters $A, B$ simultaneously by letting $\vec{\psi} = (\vec{\psi}^A, \vec{\psi}^B)$, $\vec{\lambda} = (\vec{\lambda}^A, \vec{\lambda}^B)$, and $\vec{\psi}_{obs} = (\vec{\psi}^A_{obs}, \vec{\psi}^B_{obs})$, then the solutions $\vec{\lambda}^A, \vec{\lambda}^B$ to the equations:

$$\sum_{\vec{x}} P(\vec{x}|\vec{\lambda}^A, \vec{\lambda}^B)(\vec{\phi}^A(\vec{x}), \vec{\phi}^B(\vec{x})) = (\vec{\psi}^A_{obs}, \vec{\psi}^B_{obs}), \quad (3)$$

are the same (approximately) as the solutions to the equations $\sum_{\vec{x}} P(\vec{x}|\vec{\lambda}^A)\vec{\phi}^A(\vec{x}) = \vec{\psi}^A_{obs}$ and $\sum_{\vec{x}} P(\vec{x}|\vec{\lambda}^B)\vec{\phi}^B(\vec{x}) = \vec{\psi}^B_{obs}$, see figure (3) for an example.

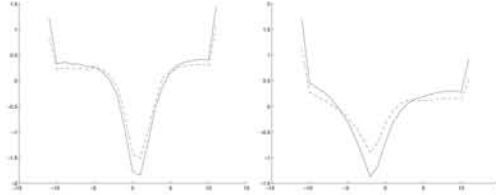

Figure 3: Evidence for decoupling of features. The left and right panels show the clique potentials learned for the features $\partial/\partial x$ and $\partial/\partial y$ respectively. The solid lines give the potentials when they are learned individually. The dashed lines show the potentials when they are learned simultaneously. Figure courtesy of Prof. Xiuwen Liu, Florida State University.

We now show how this decoupling property arises naturally if the $g$-factor for the two filters factorizes. This factorization, of course, is a property only of the form of the statistics and is *completely independent of whether the statistics of the two filters are dependent for the training data.*

Property I: *Suppose we have two sufficient statistics $\vec{\phi}^A(\vec{x}), \vec{\phi}^B(\vec{x})$ which are independent on the lattice in the sense that $g(\vec{\psi}^A, \vec{\psi}^B) = g^A(\vec{\psi}^A)g^B(\vec{\psi}^B)$, then $\log Z[\vec{\lambda}^A, \vec{\lambda}^B] = \log Z^A[\vec{\lambda}^A] + \log Z^B[\vec{\lambda}^B]$ and $\hat{P}(\vec{\psi}^A, \vec{\psi}^B) = \hat{P}^A(\vec{\psi}^A)\hat{P}^B(\vec{\psi}^B)$.*

This implies that the parameters $\vec{\lambda}^A, \vec{\lambda}^B$ can be solved from the independent equations $\frac{\partial \log Z^A[\vec{\lambda}^A]}{\partial \vec{\lambda}^A} = \vec{\psi}_{obs}^A$, $\frac{\partial \log Z^B[\vec{\lambda}^B]}{\partial \vec{\lambda}^B} = \vec{\psi}_{obs}^B$ or $\sum_{\vec{\psi}^A} \hat{P}^A(\vec{\psi}^A)\vec{\psi}^A = \vec{\psi}_{obs}^A$, $\sum_{\vec{\psi}^B} \hat{P}^B(\vec{\psi}^B)\vec{\psi}^B = \vec{\psi}_{obs}^B$.

Moreover, the resulting distribution $P(\vec{x})$ can be obtained by multiplying the distributions $(1/Z^A)e^{\vec{\lambda}^A \cdot \vec{\psi}^A(\vec{x})}$ and $(1/Z^B)e^{\vec{\lambda}^B \cdot \vec{\psi}^B(\vec{x})}$ together.

The point here is that the potential terms for the two statistics $\vec{\psi}^A, \vec{\psi}^B$ decouple if the phase factor $g(\vec{\psi}^A, \vec{\psi}^B)$ can be factorized. *We conjecture that this is effectively the case for many linear filters used in vision processing.* For example, it is plausible that the $g$-factor for features $\partial/\partial x$ and $\partial/\partial y$ factorizes – and figure (3) shows that their clique potentials do decouple (approximately). Clearly, if factorization between filters occurs then it gives great simplification to the system.

## 4   Approximating the $g$-factor for a Single Histogram

We now consider the case where the statistic is a single histogram. Our aim is to understand why features whose histograms are of stereotypical shape give rise to potentials of the form given by figure (3). Our results, of course, can be directly extended to multiple histograms if the filters decouple, see subsection (3.1). We first describe the approximation and then discuss its relevance for filter pursuit.

We rescale the $\vec{\lambda}$ variables by $N$ so that we have:

$$P(\vec{x}|\lambda) = \frac{e^{N\vec{\lambda}\cdot\vec{\phi}(\vec{x})}}{Z[\vec{\lambda}]}, \quad \hat{P}(\vec{\psi}|\lambda) = g(\vec{\psi})\frac{e^{N\vec{\lambda}\cdot\vec{\psi}}}{Z[\vec{\lambda}]}, \tag{4}$$

We now consider the approximation that the filter responses $\{f_i\}$ are *independent of each other when the images are uniformly distributed.* This is the *multinomial approximation.* (We attempted a related approximation [1] which was less successful.) It implies that we can express the phase factor as being proportional to a multinomial distribution:

$$g(\vec{\psi}) = L^N \frac{N!}{(N\psi_1)!...(N\psi_Q)!}\alpha_1^{N\psi_1}...\alpha_Q^{N\psi_Q}, \quad \hat{P}_0(\vec{\psi}) = \frac{N!}{(N\psi_1)!...(N\psi_Q)!}\alpha_1^{N\psi_1}...\alpha_Q^{N\psi_Q}, \tag{5}$$

where $\sum_{a=1}^Q \psi_a = 1$ (by definition) and the $\{\alpha_a\}$ are the means of the components $\{\psi_a\}$ with respect to the distribution $\hat{P}_0(\vec{\psi})$. As we will describe later, the $\{\alpha_a\}$ will be determined by the filters $\{f_i\}$. See Coughlan and Yuille, in preparation, for details of how to compute the $\{\alpha_a\}$.

This approximation enables us to calculate MEL *analytically.*

**Theorem** *With the multinomial approximation the log partition function is:*

$$\log Z[\vec{\lambda}] = N \log L + N \log\{\sum_{a=1}^Q e^{\lambda_a + \log \alpha_a}\}, \tag{6}$$

*and the "potentials" $\{\lambda_a\}$ can be solved in terms of the observed data $\{\psi_{obs,a}\}$ to be:*

$$\lambda_a = \log \frac{\psi_{obs,a}}{\alpha_a}, \quad a = 1, ..., Q. \tag{7}$$

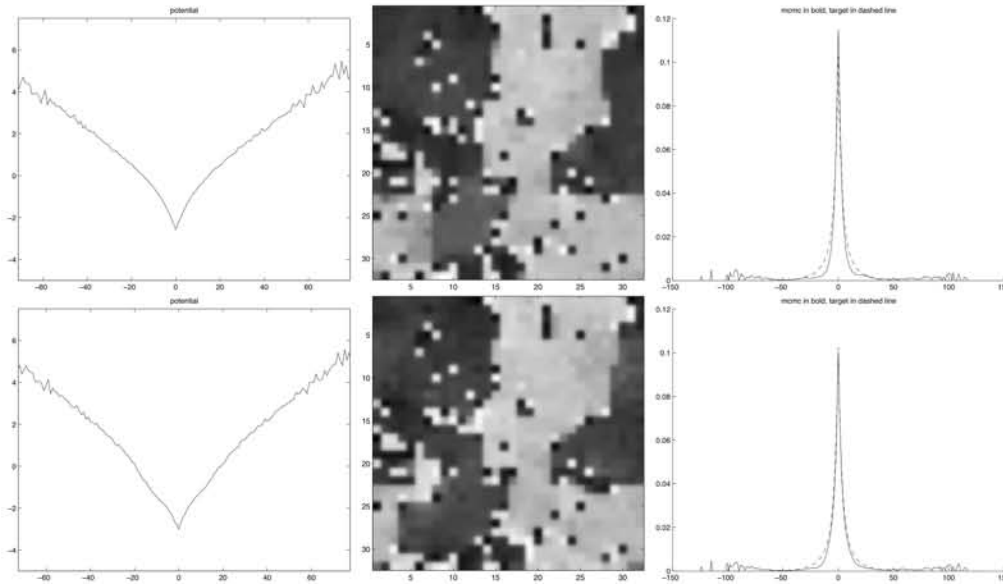

Figure 4: Top row: the multinomial approximation. Bottom row: full implementation of MEL (see text). (Left panels) the potentials, (center panels) synthesized images, and (right panels) the difference between the observed histogram (dashed line) and the histogram of the synthesized images (bold line). Filters were $d/dx$ and $d/dy$.

We note that there is an ambiguity $\lambda_a \mapsto \lambda_a + K$ where $K$ is an arbitrary number (recall that $\sum_{a=1}^{Q} \psi(a) = 1$). We fix this ambiguity by setting $\vec{\lambda} = 0$ if $\vec{\alpha} = \vec{\psi}_{obs}$.

Proof. *Direct calculation.*

Our simulation results show that this simple approximation gives the typical potential forms generated by Markov Chain Monte Carlo (MCMC) algorithms for Minimax Entropy Learning. Compare the multinomial approximation results with those obtained from a full implementation of MEL by the algorithm used in [11], see figure (4).

Filter pursuit is required to determine which filters carry most information. MEL [11] prefers filters (statistics) which give rise to low entropy distributions (this is the "Min" part of Minimax). The entropy is given by $H(P) = -\sum_{\vec{x}} P(\vec{x}|\vec{\lambda}) \log P(\vec{x}|\vec{\lambda}) = \log Z[\vec{\lambda}] - \sum_{a=1}^{Q} \lambda_a \psi_a$. For the multinomial approximation this can be computed to be $N \log L - N \sum_{a=1}^{Q} \psi_a \log \frac{\psi_a}{\alpha_a}$. This gives an intuitive interpretation of feature pursuit: we should prefer filters whose statistical response to the image training data is *as large as possible* from their responses to uniformly distributed images. This is measured by the Kullback-Leibler divergence $\sum_{a=1}^{Q} \psi_a \log \frac{\psi_a}{\alpha_a}$. Recall that if the multinomial approximation is used for multiple filters then we should simply add together the entropies of different filters.

# 5 Connections to Generalized Iterative Scaling

In this section we demonstrate a connection between the multinomial approximation and Generalized Iterative Scaling (GIS)[2]. GIS is an iterative procedure for calculating clique potentials that is guaranteed to converge to the maximum likelihood values of the potentials given the desired empirical filter marginals (e.g. filter histograms). We show that estimating the potentials by the multinomial approximation is equivalent to the estimate obtained after performing *the first iteration* of GIS. We also outline an efficient procedure that allows us to continue additional GIS iterations to improve upon the multinomial approximation.

The GIS procedure calculates a sequence of distributions on the entire image (and is guaranteed to converge to the correct maximum likelihood distribution), with an update rule given by $P^{(t+1)}(\vec{x}) \propto P^{(0)}(\vec{x}) \prod_{a=1}^{Q} \{\frac{\psi_a^{obs}}{\psi_a^{(t)}}\}^{\phi_a(\vec{x})}$, where $\psi_a^{(t)} = < \phi_a(\vec{x}) >_{P^{(t)}(\vec{x})}$ is the expected histogram for the distribution at time $t$. This implies that the corresponding clique potential update equation is given by: $\lambda_a^{(t+1)} = \lambda_a^{(t)} + \log \psi_a^{obs} - \log \psi_a^{(t)}$.

If we initialize GIS so that the initial distribution is the uniform distribution, i.e. $P^{(0)}(\vec{x}) = L^{-N}$, then the distribution after one iteration is $P^{(1)}(\vec{x}) \propto e^{\sum_a \phi_a(\vec{x}) \log(\psi_a^{obs}/\alpha_a)}$. In other words, the distribution after one iteration is the MEL distribution *with clique potential given by the multinomial approximation*. (The result can be adapted to the case of multiple filters, as explained in Coughlan and Yuille, in preparation.)

We can iterate GIS to improve the estimate of the clique potentials beyond the accuracy of the multinomial approximation. The main difficulty lies in estimating $\psi_a^{(t)}$ for $t > 0$ (at $t = 0$ this expectation is just the mean histogram with respect to the uniform distribution, $\alpha_a$, which may be calculated efficiently as described in Coughlan and Yuille, in preparation). One way to approximate these expectations is to apply a Bethe-Kikuchi approximation technique [8], used for estimating marginals on Markov Random Fields, to our MEL distribution. Our technique, which was inspired by the Unified Propagation and Scaling Algorithm [7], consists of writing the Bethe free energy [8] for our 2-d image lattice, simplifying it using the shift invariance of the lattice (which enables the algorithm to run swiftly), and using the Convex-Concave Procedure (CCCP) [9] procedure to obtain an iterative update equation to estimate the histogram expectations. The GIS algorithm is then run using these histogram expectations (the results were accurate and did not improve appreciably by using the higher-order Kikuchi free energy approximation). See Coughlan and Yuille, in preparation, for details of this procedure.

# 6 Discussion

This paper describes the $g$-factor, which depends on the lattice and quantization and is independent of the training image data. Alternatively it can be thought of as being proportional to the distribution of feature responses when the input images are uniformly distributed.

We showed that the $g$-factor can be used to relate probability distributions on features to distributions on images. In particular, we described approximations

which, when valid, enable MEL to be computed analytically. In addition, we can determine when the clique potentials for features decouple, and evaluate how informative each feature is. Finally, we establish a connection between the multinomial approximation and GIS, and outline an efficient procedure based on Bethe-Kikuchi approximations that allows us to continue additional GIS iterations to improve upon the multinomial approximation.

## Acknowledgements

We would like to thank Michael Jordan and Yair Weiss for introducing us to Generalized Iterative Scaling and related algorithms. We also thank Anand Rangarajan, Xiuwen Liu, and Song Chun Zhu for helpful conversations. Sabino Ferreira gave useful feedback on the manuscript. This work was supported by the National Institute of Health (NEI) with grant number RO1-EY 12691-01.

## References

[1] J.M. Coughlan and A.L. Yuille. "A Phase Space Approach to Minimax Entropy Learning and The Minutemax approximation". In *Proceedings NIPS'98*. 1998.

[2] J. N. Darroch and D. Ratcliff. "Generalized Iterative Scaling for Log-Linear Models". The Annals of Mathematical Statistics. 1972. Vol. 43, No. 5, 1470-1480.

[3] C. Domb and M.S. Green (Eds). **Phase Transitions and Critical Phenomena**. Vol. 2. Academic Press. London. 1972.

[4] S. M. Konishi, A.L. Yuille, J.M. Coughlan and Song Chun Zhu. "Fundamental Bounds on Edge Detection: An Information Theoretic Evaluation of Different Edge Cues." In *Proceedings Computer Vision and Pattern Recognition CVPR'99*. Fort Collins, Colorado. June 1999.

[5] A.B. Lee, D.B. Mumford, and J. Huang. "Occlusion Models of Natural Images: A Statistical Study of a Scale-Invariant Dead Leaf Model". *International Journal of Computer Vision*. Vol. 41, No.'s 1/2. January/February 2001.

[6] J. Portilla and E. P. Simoncelli. "Parametric Texture Model based on Joint Statistics of Complex Wavelet Coefficients". *International Journal of Computer Vision*. October 2000.

[7] Y. W. Teh and M. Welling. "The Unified Propagation and Scaling Algorithm." In *Proceedings NIPS'01*. 2001.

[8] J.S. Yedidia, W.T. Freeman, Y. Weiss, "Generalized Belief Propagation." In *Proceedings NIPS'00*. 2000.

[9] A.L. Yuille. "CCCP Algorithms to Minimize the Bethe and Kikuchi Free Energies," Neural Computation. In press. 2002.

[10] S.C. Zhu and D. Mumford. "Prior Learning and Gibbs Reaction-Diffusion." PAMI vol.19, no.11, pp1236-1250, Nov. 1997.

[11] S.C. Zhu, Y. Wu, and D. Mumford. "Minimax Entropy Principle and Its Application to Texture Modeling". Neural Computation. Vol. 9. no. 8. Nov. 1997.
